# SMEM Algorithm for Mixture Models

**Naonori Ueda** **Ryohei Nakano**
{ueda, nakano}@cslab.kecl.ntt.co.jp
NTT Communication Science Laboratories
Hikaridai, Seika-cho, Soraku-gun, Kyoto 619-0237 Japan

**Zoubin Ghahramani** **Geoffrey E. Hinton**
zoubin@gatsby.ucl.ac.uk g.hinton@ucl.ac.uk
Gatsby Computational Neuroscience Unit, University College London
17 Queen Square, London WC1N 3AR, UK

## Abstract

We present a split and merge EM (SMEM) algorithm to overcome the local maximum problem in parameter estimation of finite mixture models. In the case of mixture models, non-global maxima often involve having too many components of a mixture model in one part of the space and too few in another, widely separated part of the space. To escape from such configurations we repeatedly perform simultaneous split and merge operations using a new criterion for efficiently selecting the split and merge candidates. We apply the proposed algorithm to the training of Gaussian mixtures and mixtures of factor analyzers using synthetic and real data and show the effectiveness of using the split and merge operations to improve the likelihood of both the training data and of held-out test data.

## 1 INTRODUCTION

Mixture density models, in particular normal mixtures, have been extensively used in the field of statistical pattern recognition [1]. Recently, more sophisticated mixture density models such as mixtures of latent variable models (*e.g.,* probabilistic PCA or factor analysis) have been proposed to approximate the underlying data manifold [2]-[4]. The parameter of these mixture models can be estimated using the EM algorithm [5] based on the maximum likelihood framework [3][4]. A common and serious problem associated with these EM algorithm is the local maxima problem. Although this problem has been pointed out by many researchers, the best way to solve it in practice is still an open question.

Two of the authors have proposed the deterministic annealing EM (DAEM) algorithm [6], where a modified posterior probability parameterized by *temperature* is derived to avoid local maxima. However, in the case of mixture density models, local maxima arise when there are too many components of a mixture models in one part of the space and too few in another. It is not possible to move a component from the overpopulated region to the underpopulated region without passing

through positions that give lower likelihood. We therefore introduce a discrete move that simultaneously merges two components in an overpopulated region and splits a component in an underpopulated region.

The idea of split and merge operations has been successfully applied to clustering or vector quantization (*e.g.*, [7]). To our knowledge, this is the first time that simultaneous split and merge operations have been applied to improve mixture density estimation. New criteria presented in this paper can efficiently select the split and merge candidates. Although the proposed method, unlike the DAEM algorithm, is limited to mixture models, we have experimentally comfirmed that our split and merge EM algorithm obtains better solutions than the DAEM algorithm.

## 2  Split and Merge EM (SMEM) Algorithm

The probability density function (pdf) of a mixture of M density models is given by

$$p(\boldsymbol{x}; \Theta) = \sum_{m=1}^{M} \alpha_m p(\boldsymbol{x}|\omega_m; \theta_m), \quad \text{where} \quad \alpha_m \geq 0 \quad \text{and} \quad \sum_{m=1}^{M} \alpha_m = 1. \quad (1)$$

The $p(\boldsymbol{x}|\omega_m; \theta_m)$ is a $d$-dimensional density model corresponding to the component $\omega_m$. The EM algorithm, as is well known, iteratively estimates the parameters $\Theta = \{(\alpha_m, \theta_m), \ m = 1, \ldots, M\}$ using two steps. The E-step computes the expectation of the complete data log-likelihood.

$$Q(\Theta|\Theta^{(t)}) = \sum_{\boldsymbol{x}} \sum_{m} P(\omega_m|\boldsymbol{x}; \Theta^{(t)}) \log \alpha_m p(\boldsymbol{x}|\omega_m; \theta_m), \quad (2)$$

where $P(\omega_m|\boldsymbol{x}; \Theta^{(t)})$ is the posterior probability which can be computed by

$$P(\omega_m|\boldsymbol{x}; \Theta^{(t)}) = \frac{\alpha_m^{(t)} p(\boldsymbol{x}|\omega_m; \theta_m^{(t)})}{\sum_{m'=1}^{M} \alpha_{m'}^{(t)} p(\boldsymbol{x}|\omega_{m'}; \theta_{m'}^{(t)})}. \quad (3)$$

Next, the M-step maximizes this $Q$ function with respect to $\Theta$ to estimate the new parameter values $\Theta^{(t+1)}$.

Looking at (2) carefully, one can see that the $Q$ function can be represented in the form of a direct sum; *i.e.*, $Q(\Theta|\Theta^{(t)}) = \sum_{m=1}^{M} q_m(\Theta|\Theta^{(t)})$, where $q_m(\Theta|\Theta^{(t)}) = \sum_{\boldsymbol{x} \in \mathcal{X}} P(\omega_m|\boldsymbol{x}; \Theta^{(t)}) \log \alpha_m p(\boldsymbol{x}|\omega_m; \theta_m)$ and depends only on $\alpha_m$ and $\theta_m$. Let $\Theta^*$ denote the parameter values estimated by the usual EM algorithm. Then after the EM algorithm has converged, the $Q$ function can be rewritten as

$$Q^* = q_i^* + q_j^* + q_k^* + \sum_{m, m \neq i, j, k} q_m^*. \quad (4)$$

We then try to increase the first three terms of the right-hand side of (4) by merging two components $\omega_i$ and $\omega_j$ to produce a component $\omega_{i'}$, and splitting the component $\omega_k$ into two components $\omega_{j'}$ and $\omega_{k'}$. To reestimate the parameters of these new components, we have to initialize the parameters corresponding to them using $\Theta^*$.

The initial parameter values for the merged component $\omega_{i'}$ can be set as a linear combination of the original ones before merge:

$$\alpha_{i'} = \alpha_i^* + \alpha_j^* \quad \text{and} \quad \theta_{i'} = \frac{\theta_i^* \sum_{\boldsymbol{x}} P(\omega_i|\boldsymbol{x}; \Theta^*) + \theta_j^* \sum_{\boldsymbol{x}} P(\omega_j|\boldsymbol{x}; \Theta^*)}{\sum_{\boldsymbol{x}} P(\omega_i|\boldsymbol{x}; \Theta^*) + \sum_{\boldsymbol{x}} P(\omega_j|\boldsymbol{x}; \Theta^*)}. \quad (5)$$

On the other hand, as for two components $\omega_{j'}$ and $\omega_{k'}$, we set

$$\alpha_{j'} = \alpha_{k'} = \alpha_k^*/2 \qquad \theta_{j'} = \theta_k^* + \epsilon \quad \text{and} \quad \theta_{k'} = \theta_k^* + \epsilon', \tag{6}$$

where $\epsilon$ is some small random perturbation vector or matrix ($i.e.,$ $\|\epsilon\| \ll \|\theta_k^*\|$)[1]. The parameter reestimation for $m = i', j'$ and $k'$ can be done by using EM steps, but note that the posterior probability (3) should be replaced with (7) so that this reestimation does not affect the other components.

$$P(\omega_m|\boldsymbol{x};\Theta^{(t)}) = \frac{\alpha_m^{(t)} p(\boldsymbol{x}|\omega_m;\theta_m^{(t)})}{\displaystyle\sum_{m'=i',j',k'} \alpha_{m'}^{(t)} p(\boldsymbol{x}|\omega_m;\theta_{m'}^{(t)})} \times \sum_{m'=i,j,k} P(\omega_{m'}|\boldsymbol{x};\Theta^*), \quad m = i',j',k'.$$

$$\tag{7}$$

Clearly $\sum_{m'=i',j',k'} P(\omega_{m'}|\boldsymbol{x};\Theta^{(t)}) = \sum_{m=i,j,k} P(\omega_m|\boldsymbol{x};\Theta^*)$ always holds during the reestimation process. For convenience, we call this EM procedure the *partial EM procedure*. After this partial EM procedure, the usual EM steps, called the *full EM procedure*, are performed as a post processing. After these procedures, if $Q$ is improved, then we accept the new estimate and repeat the above after setting the new paramters to $\Theta^*$. Otherwise reject and go back to $\Theta^*$ and try another candidate. We summarize these procedures as follows:

**[SMEM Algorithm]**

1. Perform the usual EM updates. Let $\Theta^*$ and $Q^*$ denote the estimated parameters and corresponding $Q$ function value, respectively.
2. Sort the split and merge candidates by computing split and merge criteria (described in the next section) based on $\Theta^*$. Let $\{i,j,k\}_c$ denote the $c$th candidate.
3. For $c = 1, \ldots, C_{max}$, perform the following: After initial parameter settings based on $\Theta^*$, perform the *partial EM procedure* for $\{i,j,k\}_c$ and then perform the *full EM procedure*. Let $\Theta^{**}$ be the obtained parameters and $Q^{**}$ be the corresponding $Q$ function value. If $Q^{**} > Q^*$, then set $Q^* \leftarrow Q^{**}$, $\Theta^* \leftarrow \Theta^{**}$ and go to Step 2.
4. Halt with $\Theta^*$ as the final parameters.

Note that when a certain split and merge candidate which improves the $Q$ function value is found at Step 3, the other successive candidates are ignored. There is therefore no guarantee that the split and the merge candidates that are chosen will give the largest possible improvement in $Q$. This is not a major problem, however, because the split and merge operations are performed repeatedly. Strictly speaking, $C_{max} = M(M-1)(M-2)/2$, but experimentally we have confirmed that $C_{max} \sim 5$ may be enough.

## 3  Split and Merge Criteria

Each of the split and merge candidates can be evaluated by its $Q$ function value after Step 3 of the SMEM algorithm mentioned in Sec.2. However, since there are so many candidates, some reasonable criteria for ordering the split and merge candidates should be utilized to accelerate the SMEM algorithm.

In general, when there are many data points each of which has almost equal posterior probabilities for any two components, it can be thought that these two components

might be merged. To numerically evaluate this, we define the following merge criterion:

$$J_{merge}(i, j; \Theta^*) = \mathbf{P}_i(\Theta^*)^T \mathbf{P}_j(\Theta^*), \tag{8}$$

where $\mathbf{P}_i(\Theta^*) = (P(\omega_i|\boldsymbol{x}_1; \Theta^*), \ldots, P(\omega_i|\boldsymbol{x}_N; \Theta^*))^T \in \mathcal{R}^N$ is the $N$-dimensional vector consisting of posterior probabilities for the component $\omega_i$. Clearly, two components $\omega_i$ and $\omega_j$ with larger $J_{merge}(i, j; \Theta^*)$ should be merged.

As a split criterion ($J_{split}$), we define the *local Kullback divergence* as:

$$J_{split}(k; \Theta^*) = \int p_k(\boldsymbol{x}; \Theta^*) \log \frac{p_k(\boldsymbol{x}; \Theta^*)}{p(\boldsymbol{x}|\omega_k; \theta_k^*)} d\boldsymbol{x}, \tag{9}$$

which is the distance between two distributions: the local data density $p_k(\boldsymbol{x})$ around the component $\omega_k$ and the density of the component $\omega_k$ specified by the current parameter estimate $\boldsymbol{\mu}_k^*$ and $\Sigma_k^*$. The local data density is defined as:

$$p_k(\boldsymbol{x}; \Theta^*) = \frac{\sum_{n=1}^N \delta(\boldsymbol{x} - \boldsymbol{x}_n) P(\omega_k|\boldsymbol{x}_n; \Theta^*)}{\sum_{n=1}^N P(\omega_k|\boldsymbol{x}_n; \Theta^*)}. \tag{10}$$

This is a modified empirical distribution weighted by the posterior probability so that the data around the component $\omega_k$ are focused. Note that when the weights are equal, *i.e.*, $P(\omega_k|\boldsymbol{x}; \Theta^*) = 1/M$, (10) is the usual empirical distribution, *i.e.*, $p_k(\boldsymbol{x}; \Theta^*) = (1/N) \sum_{n=1}^N \delta(\boldsymbol{x} - \boldsymbol{x}_n)$. Since it can be thought that the component with the largest $J_{split}(k; \Theta^*)$ has the worst estimate of the local density, we should try to split it. Using $J_{merge}$ and $J_{split}$, we sort the split and merge candidates as follows. First, merge candidates are sorted based on $J_{merge}$. Then, for each sorted merge candidate $\{i, j\}_c$, split candidates excluding $\{i, j\}_c$ are sorted as $\{k\}_c$. By combining these results and renumbering them, we obtain $\{i, j, k\}_c$.

## 4 Experiments

### 4.1 Gaussian mixtures

First, we show the results of two-dimensional synthetic data in Fig. 1 to visually demonstrate the usefulness of the split and merge operations. Initial mean vectors and covariance matrices were set to near mean of all data and unit matrix, respectively. The usual EM algorithm converged to the local maximum solution shown in Fig. 1(b), whereas the SMEM algorithm converged to the superior solution shown in Fig. 1(d) very close to the true one. The split of the 1st Gaussian shown in Fig. 1(c) seems to be redundant, but as shown in Fig. 1(d) they are successfully merged and the original two Gaussians were improved. This indicates that the split and merge operations not only appropriately assign the number of Gaussians in a local data space, but can also improve the Gaussian parameters themselves.

Next, we tested the proposed algorithm using 20-dimensional real data (facial images) where the local maxima make the optimization difficult. The data size was 103 for training and 103 for test. We ran three algorithms (EM, DAEM, and SMEM) for ten different initializations using the $K$-means algorithm. We set $M = 5$ and used a diagonal covariance for each Gaussian. As shown in Table 1, the worst solution found by the SMEM algorithm was better than the best solutions found by the other algorithms on both training and test data.

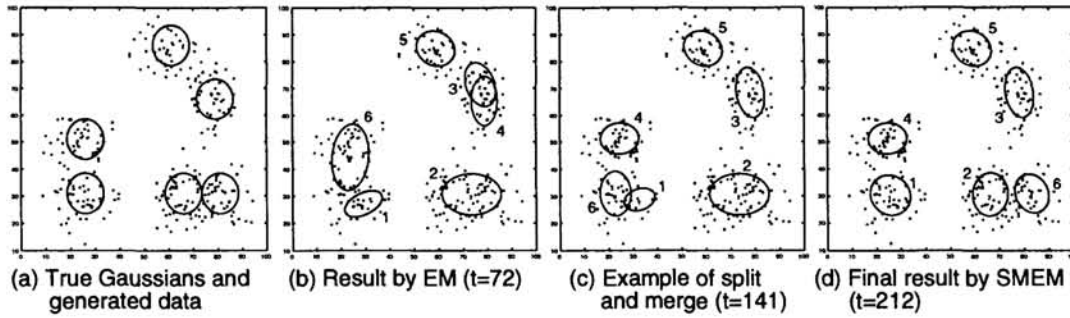

(a) True Gaussians and generated data  (b) Result by EM (t=72)  (c) Example of split and merge (t=141)  (d) Final result by SMEM (t=212)

Figure 1: Gaussian mixture estimation results.

Table 1: Log-likelihood / data point

|  |  | initiall value | EM | DAEM | SMEM |
|---|---|---|---|---|---|
| Training data | mean | −159.1 | −148.2 | −147.9 | −145.1 |
|  | std | 1.77 | 0.24 | 0.04 | 0.08 |
|  | max | −157.3 | −147.7 | −147.8 | −145.0 |
|  | min | −163.2 | −148.6 | −147.9 | −145.2 |
| Test data | mean | −168.2 | −159.8 | −159.7 | −155.9 |
|  | std | 2.80 | 1.00 | 0.37 | 0.09 |
|  | max | −165.5 | −158.0 | −159.6 | −155.9 |
|  | min | −174.2 | −160.8 | −159.8 | −156.0 |

Table 2: No. of iterations

|  | EM | DAEM | SMEM |
|---|---|---|---|
| mean | 47 | 147 | 155 |
| std | 16 | 39 | 44 |
| max | 65 | 189 | 219 |
| min | 37 | 103 | 109 |

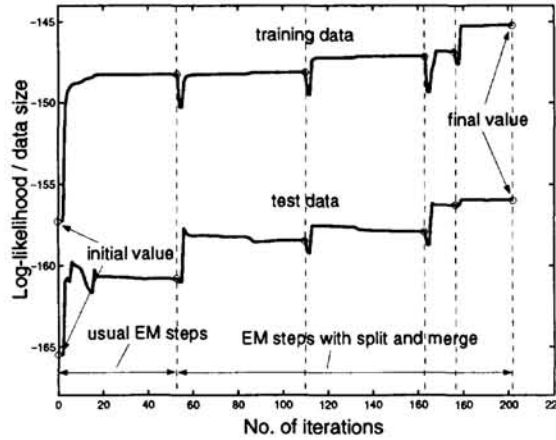

Figure 2: Trajectories of loglikelihood. Upper (lower) corresponds to training (test) data.

Figure 2 shows log-likelihood value trajectories accepted at Step 3 of the SMEM algorithm during the estimation process [2]. Comparing the convergence points at Step 3 marked by the 'o' symbol in Fig. 2, one can see that the successive split and merge operations improved the log-likelihood for both the training and test data, as we expected. Table 2 compares the number of iterations executed by the three algorithms. Note that in the SMEM algorithm, the EM-steps corresponding to rejected split and merge operations are not counted. The average rank of the accepted split and merge candidates was 1.8 (STD=0.9), which indicates that the proposed split and merge criteria work very well. Therefore, the SMEM algorithm was about $155 \times 1.8/47 \simeq 6$ times slower than the original EM algorithm.

## 4.2 Mixtures of factor analyzers

A mixture of factor analyzers (MFA) can be thought of as a reduced dimension mixture of Gaussians [4]. That is, it can extract locally linear low-dimensional manifold underlying given high-dimensional data. A single FA model assumes that an observed $D$-dimensional variable $x$ are generated as a linear transformation of some lower $K$-dimensional *latent* variable $z \sim \mathcal{N}(0, I)$ plus additive Gaussian noise $v \sim \mathcal{N}(0, \Psi)$. $\Psi$ is diagonal. That is, the generative model can be written as

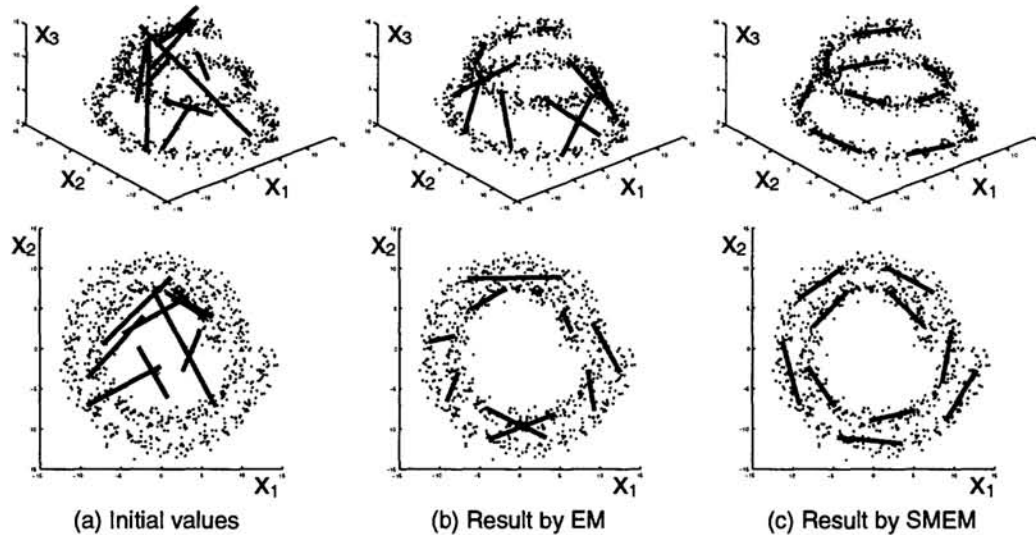

(a) Initial values          (b) Result by EM          (c) Result by SMEM

Figure 3: Extraction of 1D manifold by using a mixture of factor analyzers.

$x = \Lambda z + v + \mu$. Here $\mu$ is a mean vector. Then from simple calculation, we can see that $x \sim \mathcal{N}(\mu, \Lambda\Lambda^T + \Psi)$. Therefore, in the case of a $M$ mixture of FAs, $x \sim \sum_{m=1}^{M} \alpha_m \mathcal{N}(\mu_m, \Lambda_m\Lambda_m^T + \Psi_m)$. See [4] for the details. Then, in this case, the $Q$ function is also decomposable into $M$ components and therefore the SMEM algorithm is straightforwardly applicable to the parameter estimation of the MFA models.

Figure 3 shows the results of extracting a one-dimensional manifold from three-dimensional data (noisy shrinking spiral) using the EM and the SMEM algorithms[3]. Although the EM algorithm converged to a poor local maxima, the SMEM algorithm successfully extracted data manifold. Table 3 compares average log-likelihood per data point over ten different initializations. The log-likelihood values were drastically improved on both training and test data by the SMEM algorithm.

The MFA model is applicable to pattern recognition tasks [2][3] since once an MFA model is fitted to each class, we can compute the posterior probabilities for each data point. We tried a digit recognition task (10 digits (classes))[4] using the MFA model. The computed log-likelihood averaged over ten classes and recognition accuracy for test data are given in Table 4. Clearly, the SMEM algorithm consistently improved the EM algorithm on both log-likelihood and recognition accuracy. Note that the recognition accuracy by the 3-nearest neighbor (3NN) classifier was 88.3%. It is interesting that the MFA approach by both the EM and SMEM algorithms could outperform the nearest neighbor approach when $K = 3$ and $M = 5$. This suggests that the intrinsic dimensionality of the data would be three or so.

Table 3: Log-likelihood / data point

|  | EM | SMEM |
|---|---|---|
| Training | −7.68 (0.151) | −7.26 (0.017) |
| Test | −7.75 (0.171) | −7.33 (0.032) |

( ) : STD

Table 4: Digit recognition results

|  |  | Log-likelihood / data point | | Recognition rate (%) | |
|---|---|---|---|---|---|
|  |  | EM | SMEM | EM | SMEM |
| K=3 | M=5 | −3.18 | −3.15 | 89.0 | 91.3 |
|  | M=10 | −3.09 | −3.05 | 87.5 | 88.7 |
| K=8 | M=5 | −3.14 | −3.11 | 85.3 | 87.3 |
|  | M=10 | −3.04 | −3.01 | 82.5 | 85.1 |

## 5   Conclusion

We have shown how simultaneous split and merge operations can be used to move components of a mixture model from regions of the space in which there are too many components to regions in which there are too few. Such moves cannot be accomplished by methods that continuously move components through intermediate locations because the likelihood is lower at these locations. A simultaneous split and merge can be viewed as a way of tunneling through low-likelihood barriers, thereby eliminating many non-global optima. In this respect, it has some similarities with simulated annealing but the moves that are considered are long-range and are very specific to the particular problems that arise when fitting a mixture model. Note that the SMEM algorithm is applicable to a wide variety of mixture models, as long as the decomposition (4) holds. To make the split and merge method efficient we have introduced criteria for deciding which splits and merges to consider and have shown that these criteria work well for low-dimensional synthetic datasets and for higher-dimensional real datasets. Our SMEM algorithm consistently outperforms standard EM and therefore it would be very useful in practice.

## Footnotes

[1]In the case of mixture Gaussians, covariance matrices $\Sigma_{j'}$ and $\Sigma_{k'}$ should be positive definite. In this case, we can initialize them as $\Sigma_{j'} = \Sigma_{k'} = \det(\Sigma_k^*)^{1/d} I_d$ indtead of (6).

[2]Dotted lines in Fig. 2 denote the starting points of Step 2. Note that it is due to the initialization at Step 3 that the log-likelihood decreases just after the split and merge.

[3]In this case, each factor loading matrix $\Lambda_m$ becomes a three dimensional column vector corresponding to each thick line in Fig. 3. More correctly, the center position and the direction of each thick line are $\mu_m$ and $\Lambda_m$, respectively. And the length of each thick line is $2 \|\Lambda_m\|$.

[4]The data were created using the degenerate Glucksman's feature (16 dimensional data) by NTT labs.[8]. The data size was 200/class for training and 200/class for test.

## References

[1] MacLachlan, G. and Basford K., "Mixture models: Inference and application to clustering," Marcel Dekker, 1988.

[2] Hinton G. E., Dayan P., and Revow M., "Modeling the minifolds of images of handwritten digits," *IEEE Trans. PAMI*, vol.8, no.1, pp. 65–74, 1997.

[3] Tipping M. E. and Bishop C. M., "Mixtures of probabilistic principal component analysers," Tech. Rep. NCRG-97-3, Aston Univ. Birmingham, UK, 1997.

[4] Ghahramani Z. and Hinton G. E., "The EM algorithm for mixtures of factor analyzers," Tech. Report CRG-TR-96-1, Univ. of Toronto, 1997.

[5] Dempster A. P., Laird N. M. and Rubin D. B., "Maximum likelihood from incomplete data via the EM algorithm," *Journal of Royal Statistical Society B*, vol. 39, pp. 1–38, 1977.

[6] Ueda N. and Nakano R., "Deterministic annealing EM algorithm," *Neural Networks*, vol.11, no.2, pp.271–282, 1998.

[7] Ueda N. and Nakano R., "A new competitive learning approach based on an equidistortion principle for designing optimal vector quantizers," *Neural Networks*, vol.7, no.8, pp.1211–1227, 1994.

[8] Ishii K., "Design of a recognition dictionary using artificially distorted characters," *Systems and computers in Japan*, vol.21, no.9, pp. 669–677, 1989.
